# Learning to See Where and What: Training a Net to Make Saccades and Recognize Handwritten Characters

**Gale Martin, Mosfeq Rashid, David Chapman, and James Pittman**

MCC, 3500 Balcones Center Drive, Austin, Texas 78759

## ABSTRACT

This paper describes an approach to integrated segmentation and recognition of hand–printed characters. The approach, called *Saccade*, integrates ballistic and corrective saccades (eye movements) with character recognition. A single backpropagation net is trained to make a classification decision on a character centered in its input window, as well as to estimate the distance of the current and next character from the center of the input window. The net learns to accurately estimate these distances regardless of variations in character width, spacing between characters, writing style and other factors. During testing, the system uses the net–extracted classification and distance information, along with a set of jumping rules, to jump from character to character.

The ability to read rests on multiple foundation skills. In learning how to read, people learn how to recognize individual characters centered in the visual field. They also learn how to move their eyes along a line of text, sequentially centering the visual field on successive characters. We believe that the key to developing optical character recognition (OCR) systems that can mimic human reading capabilities, is to develop systems that can learn these and other skills in an integrated fashion. In this paper, we demonstrate that a backpropagation net can learn to navigate along a line of handwritten characters, as well as to recognize the characters centered in its visual field. The system, called *Saccade*, extends the current state of the art in OCR technology by using a single classifier to accurately and efficiently locate and recognize characters, regardless of whether they touch each other or are separate. The *Saccade* system was described briefly at the last NIPS conference (Martin & Rashid, 1992). In this paper, we describe it more fully and report on results demonstrating its accuracy and efficiency in recognizing handwritten digits.

The *Saccade* system takes a cue from the ballistic and corrective saccades (eye movements) of natural vision systems. Natural saccades make it possible to efficiently move from one informative area to another by jumping. The eye typically initiates a ballistic saccade to

move the center of focus to the general area of interest, followed, if necessary, by one or more corrective saccades for fine–grained position corrections. Recognition processes are applied only at these multiple fixation points.

We have copied some of these aspects in the artificial *Saccade* system by training a neural network to know about the locations of characters in its input window, as well as to know about the identity of the character centered in its input window. During run–time, the *Saccade* system accesses this information computed by the net for successive input windows, along with a set of simple jumping rules, to yield an OCR system that jumps from character to character, classifying each character in a sequence.

## 1   TRAINING DETAILS

As shown in Figure 1, the Saccade system has a wide input window, large enough to contain

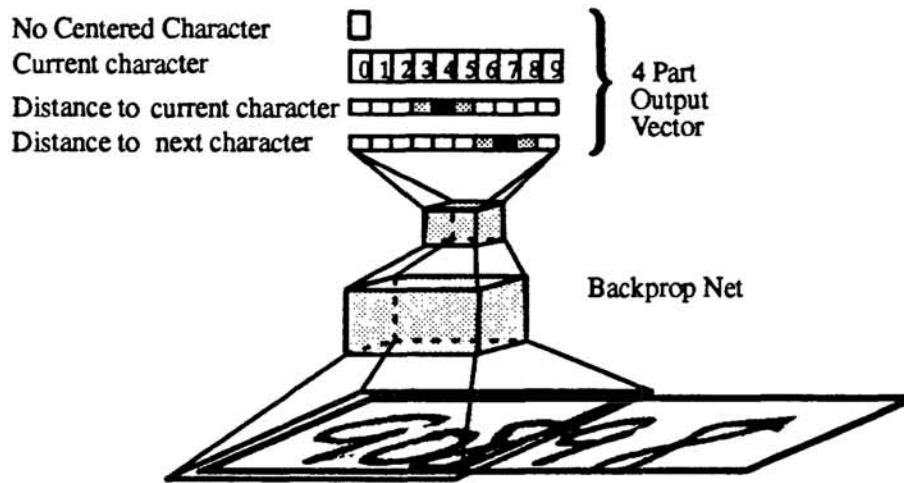

Figure 1.    The *Saccade* system uses an enlarged input window and a 4–part output vector.

several characters. Prior to training, each field image of a line of characters is labeled with the horizontal center position of each character in the field, as well as with the category of each character. During training, the input window slides horizontally across a field of characters, and at each position, the contents of the input window are paired with a four–part target output vector, the values of which are computed from the labeled information. The target values answer the following four questions about the contents of the input window:

1.   Is a character centered in the input window?
2.   What character is closest to the center of the window?
3.   How far off–center (horizontally) is the centermost character?
4.   How far is the next character to the right from the center of the window?

The first node in the output vector represents the *no–centered–character* state. It's target value is set high (e.g., 1.0) when the center of the input window falls between characters,

and set low (e.g., 0.0) when the center of the input window falls on the center of a character. When the net is trained, the value of the *no-centered-character* node indicates whether the input window is centered over a character, or whether a corrective saccade is needed to better center the character.

The second part of the output vector contains a node for each character category. When the center of the input window falls on a character, the target value for its corresponding node is set high; otherwise it is set low. When the net is trained, the values in these nodes are used to classify the centered character. The target values for both the *no-centered-character* and the *character-category* nodes are defined continuously across the horizontal dimension as trapezoidal functions, such that there are plateaus surrounding the off and on positions, with linearly increasing and decreasing values connecting the plateaus.

The third and fourth components of the output vector represent distance values, each encoded in a distributed fashion across multiple nodes, using localized receptive fields (Moody & Darken, 1988). The first of these two parts represents the distance by which the character closest to the center of the window is off-center. The target value can be positive, indicating that the center of the window is to the left of the center of character, or it can be negative, indicating that the center of the window has passed over the character, to the right of it's center. When trained, the value of the *current-character-distance* set of nodes is accessed to determine the magnitude of a corrective saccade, to make a fine-grained position adjustment.

The fourth component represents the distance from the center of the window to the center of the next character to the right. The target value can only be positive. When trained, the value of this set of nodes is accessed to determine the magnitude of a ballistic saccade, to jump to the next character to the right.

It is important to note that for both distance components, the maximum target value can not exceed half the window width. The net is never trained to make a distance judgment that extends beyond its field of view, since it is not given any information about what exists outside of it's input window. For example, when the center of the next character to the right is positioned outside of the current input window, the distance value is set to the maximum value of half the window width. Since the distance values vary with different characters, different writers, and of course, at different positions with respect to a character, the net is forced to learn to use the visual characteristics particular to each window to estimate the distance values. In other words, the net does NOT simply learn average values for each of the two distance metrics. Moreover, as the results will show, the trained net does not seem to use simple density histogram cues to estimate the distance values. It is able to reliably estimate the distance values even when characters overlap, and hence would appear as a single clump in a density histogram.

## 2   RUN-TIME SACCADE RULES

During run-time, the labeled values are, of course, not available. The system uses the computed values in the character classification and distance components of the output vector, and some heuristics, to navigate horizontally along a character field, jumping from one character to the next, and occasionally making a corrective saccade to improve its ability to classify a character. When the net recognizes a character, it executes a ballistic saccade

to the next character, obtaining the distance to jump by reading the *next–character–distance* component of the output vector. When this action fails to center a character, as indicated by a low value in the *no–centered–character* output node, the system executes a corrective saccade to better center the character. It obtains the distance and direction to jump by reading the *current–character–distance* component of the output vector. Multiple corrective saccades can be executed.

## 3   TESTING ON NIST HANDWRITTEN DIGIT FIELDS

We tested the performance of the system on a set of hand–printed digits collected and distributed by the National Institute of Standards and Technology (NIST). This is a database containing 273,000 samples of handwritten numerals. Each of 2100 Census workers filled in a form with 33 fields, 28 fields of which only contain handwritten digits. The scanning resolution of the samples was 300 pixels/inch. The neural net was trained on about 80,000 characters from 20,000 fields, written by 800 different individuals. The fields varied in length from 2 characters per field to 6 characters per field. The horizontal positions of each of the characters in these training–data fields were extracted by a person. The test data contained about 20,000 digits from 5,000 fields, written by a different group of 200 individuals. The test set was chosen to be this large because use of smaller test sets (e.g., 5,000 digits, 1250 fields) yielded significant between–set variations in reported accuracy. Each field image was preprocessed to remove the box around the field of characters, and any surrounding white space. Each field image was size normalized, with respect to the vertical axis, to a height of 20 pixels. Aspect ratio was maintained. An input pattern generator was then passed over the field to create input windows for training the net. The input window size was 36 pixels wide and 20 pixels high. The input window scanned the field at 2–pixel increments during training. Subsequent experiments have shown that training can be speeded up considerably by training on the character centers and at random points between the character centers, without causing decreased accuracy.

The backpropagation network architecture is described more fully in Martin & Rashid (1992). It has 2 hidden layers with local, shared connections in the first hidden layer, and local connections in the second hidden layer. **Shared weights are not used in the second hidden layer** because early experiments showed that this retards learning, presumably because extending the position invariance to second–hidden–layer nodes inhibits the net in learning the position specific information regarding what is centered in its input window. The learning rate of the net was initially set at .05, and then successively lowered as training reached an asymptote. The momentum term was set at .9 throughout training. All nodes in the net used logistic activation functions.

Table 1 reports on the test results in terms of field–based reject rates, for 1% and .5% percent of the fields rejected. The error rates are field–based in the sense that if the net misclassifies one character in the field, the entire field is considered as mis–classified. Error rates pertain to the fields remaining after rejection. Rejections are based on placing a threshold for the acceptable distance between the highest and next highest running activation total. In this way, by varying the threshold, the error rate can be traded off against the percentage of rejections. In addition, recognized fields were also rejected if the number of recognized digits differed from the expected number of digits.

## Table 1:  Field–Based Error Rates For Saccade System

| Field Size | Field Error Rate | Field Reject Rate |
|------------|------------------|-------------------|
| 2–digits | 1.0% | 6.4% |
|          | 0.5% | 9.3% |
| 3–digits | 1.0% | 12.7% |
|          | 0.5% | 19.6% |
| 4–digits | 1.1% | 19.5% |
|          | 0.5% | 35.0% |
| 5–digits | 1.1% | 23.2% |
|          | 0.5% | 28.3% |
| 6–digits | 1.1% | 26.8% |
|          | 0.5% | 35.0 % |

Figure 2 presents some of the fields of connected characters that the system correctly recognized. Conventional OCR systems typically fail on connected characters because they employ an independent character segmentation stage, in which the character is isolated from its surround using features, such as intervening white spaces. This character segmentation stage typically fails when characters are connected. The *Saccade* system goes beyond conventional OCR systems by integrating segmentation and recognition, and thereby is able to recognize touching characters.

The *Saccade* system is also efficient in the sense that it typically jumps from one character to the next without making a corrective saccades. Corrective saccades tend to be more likely when characters are touching. In addition, there is almost always a corrective saccade for the first character in the field, since the system starts at the beginning of the field, with

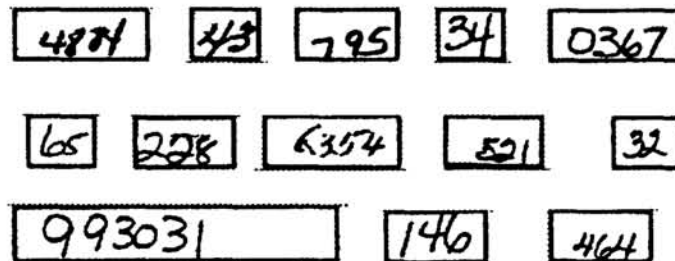

Figure 2. Examples of connected and broken characters that the *Saccade* system correctly recognizes.

no knowledge of the location of the character. For fields containing two digits, the average number of passes of the net on a small test set was about 2 saccades per character. For field containing five digits, the average number of saccades per character was 1.3.

## 4   COMPARISONS WITH OTHER SYSTEMS

The *Saccade* system is an extension of a related integrated segmentation and recognition system we reported on at last years NIPS conference (Martin & Rashid, 1992). That system employed an exhaustive scan technique, rather than saccades, to navigate along a line of text. Essentially, the net was convolved horizontally across a field image at a scan increment of 2 pixels. The net architecture was very similar to that of the *Saccade* system, except that it did not have the two distance components in the output vector. The accuracy rates of the two systems are essentially equivalent. However, the exhaustive scan version was considerably less efficient, requiring a forward pass of the network at every 2–pixel incremental scan position. On average it required about 5.5 forward passes per character, rather than the 1.3 forward passes per character required by the *Saccade* system.

Over the past two years, an approach similar to the exhaustive scan method has been advanced by a number of researchers (Keeler & Rumelhart, 1992; Matan, Burges, Le Cun, & Denker, 1992). This approach also involves convolving a network across a field image, but uses a time–delay–neural–net (TDNN), or completely local, shared weight, architecture, a smaller input window, and no explicit position labeling of characters. The TDNN approach has algorithmic advantages over the exhaustive scan version described in the previous paragraph, because the completely shared–weight architecture enables the number of forward passes of the net to be reduced considerably.

## 5   CONCLUSIONS AND FUTURE WORK

As stated at the beginning of this paper, we believe that the key to developing optical character recognition (OCR) systems that can mimic human reading capabilities is to develop systems that can learn the multiple foundation skills underlying human reading. This paper has reported some progress in this regard. We have demonstrated that a relatively simple backpropagation network can integrate its learning of position and category information, thereby enabling efficient navigation along a field of text through ballistic and corrective saccades, and accurate recognition of touching or broken characters.

There is however, a long way to go before we can claim a system with capabilities similar to human reading. The present *Saccade* system only moves horizontally, in one dimension. Human reading operates in two–dimensions, and in a sense, it operates in three–dimensions because it automatically operates across different scales. Human vision also employs automatic contrast adjustment; the *Saccade* system does not. Human vision has a wider field of view and employs a foveal transform, such that objects centered in the field of vision are represented at a higher resolution than objects in the periphery. This effectively expands the field of vision beyond what would be estimated simply by the size of the receptive area on the retina. As a result, saccades enable very effective means of scanning a large visual area. The present artificial *Saccade* system has only a small field of vision, and no foveal transform, so it's saccades must necessarily be limited in size. The present system is also only oriented toward recognizing a single character centered in its input window at

a time. Human reading typically only makes one or two saccades per word. Finally, human reading capabilities clearly integrate recognition processes with higher–level processes, to enable the redundancies of natural language to constrain the recognition decisions.

## References

Keeler, J, & Rumelhart, D. E. (1992) A self–organizing integrated segmentation and recognition neural network. In Moody, J.E., Hanson, S.J., and Lippmann, R.P., (eds.) *Advances in Neural Information Processing Systems 4*. San Mateo, CA:  Morgan Kaufmann Publishers.

Matan, O., Burges, J. C., Le Cun, Y., and Denker, J. S. (1992) Multi–Digit Recognition Using a Space Displacement Neural Network. In Moody, J.E., Hanson, S.J., and Lippmann, R.P., (eds.) *Advances in Neural Information Processing Systems 4*. San Mateo, CA:  Morgan Kaufmann Publishers, 488–495.

Martin, G. L. & Rashid, M. (1992) Recognizing overlapping hand–printed characters by centered–object integrated segmentation and recognition. In Moody, J.E., Hanson, S.J., and Lippmann, R.P., (eds.) *Advances in Neural Information Processing Systems 4*. San Mateo, CA:  Morgan Kaufmann Publishers.

Moody, J. & Darken, C. (1988) Learning with localized receptive fields. Technical Report Yaleu/DCS/RR–649.
